# A Novel Reinforcement Model of Birdsong Vocalization Learning

**Kenji Doya**
ATR Human Information Processing
Research Laboratories
2-2 Hikaridai, Seika, Kyoto 619-02, Japan

**Terrence J. Sejnowski**
Howard Hughes Medical Institute
UCSD and Salk Institute,
San Diego, CA 92186-5800, USA

## Abstract

Songbirds learn to imitate a tutor song through auditory and motor learning. We have developed a theoretical framework for song learning that accounts for response properties of neurons that have been observed in many of the nuclei that are involved in song learning. Specifically, we suggest that the *anterior forebrain pathway*, which is not needed for song production in the adult but is essential for song acquisition, provides synaptic perturbations and adaptive evaluations for syllable vocalization learning. A computer model based on reinforcement learning was constructed that could replicate a real zebra finch song with 90% accuracy based on a spectrographic measure. The second generation of the birdsong model replicated the tutor song with 96% accuracy.

## 1 INTRODUCTION

Studies of motor pattern generation have generally focussed on innate motor behaviors that are genetically preprogrammed and fine-tuned by adaptive mechanisms (Harris-Warrick et al., 1992). Birdsong learning provides a favorable opportunity for investigating the neuronal mechanisms for the acquisition of complex motor patterns. Much is known about the neuroethology of birdsong and its neuroanatomical substrate (see Nottebohm, 1991 and Doupe, 1993 for reviews), but relatively little is known about the overall system from a computational viewpoint. We propose a set of hypotheses for the functions of the brain nuclei in the song system and explore their computational strength in a model based on biological constraints. The model could reproduce real and artificial birdsongs in a few hundred learning trials.

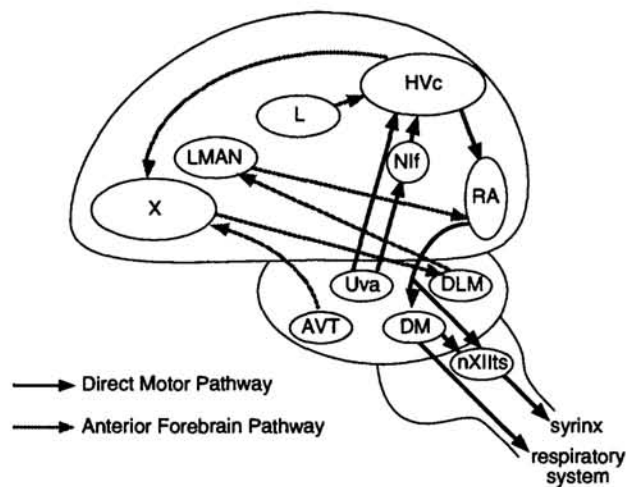

Figure 1: Major songbird brain nuclei involved in song control. The dark arrows show the direct motor control pathway and the gray arrows show the anterior forebrain pathway. Abbreviations: Uva, nucleus uvaeformis of the thalamus; NIf, nucleus interface of the neostriatum; L, field L (primary auditory are of the forebrain); HVc, higher vocal center; RA, robust nucleus of the archistriatum; DM, dorso-medial part of the nucleus intercollicularis; nXIIts, tracheosyringeal part of the hypoglossal nucleus; AVT, ventral area of Tsai of the midbrain; X, area X of lobus parolfactorius; DLM, medial part of the dorsolateral nucleus of the thalamus; LMAN, lateral magnocellular nucleus of the anterior neostriatum.

## 2   NEUROETHOLOGY OF BIRDSONG

Although songs from individual birds of the same species may sound quite similar, a young male songbird *learns* to sing by imitating the song of a tutor, which is usually the father or another adult male in the colony. If a young bird does not hear a tutor song during a *critical period*, it will sing short, poorly structured songs, and if a bird is deafened in the period when it practices vocalization, it develops highly abnormal songs. These observations indicate that there are two phases in song learning: the sensory learning phase when a young bird memorizes *song templates* and the motor learning phase in which the bird establishes the motor programs using auditory feedback (Konishi, 1965). These two phases can be separated by several months in some species, implying that birds have remarkable capability for memorizing complex temporal sequences. Once a song is *crystallized*, its pattern is very stable. Even deafening the bird has little immediate effect.

The brain nuclei involved in song learning are shown in Figure 1. The primary motor control pathway is composed of Uva, NIf, HVc, RA, DM, and nXIIts. If any of these nuclei is lesioned, a bird cannot sing normally. Experimental studies suggest that HVc is involved in generating syllable sequences and that RA produces motor commands for each syllable (Vu et al., 1994). Interestingly, neurons in HVc, RA and nXIIts show vigorous auditory responses, suggesting that the motor control system is closely coupled with the auditory system (Nottebohm, 1991).

There is also a "bypass" from HVc to RA which consists of area X, DLM, and LMAN called the *anterior forebrain pathway* (Doupe, 1993). This pathway is not directly involved

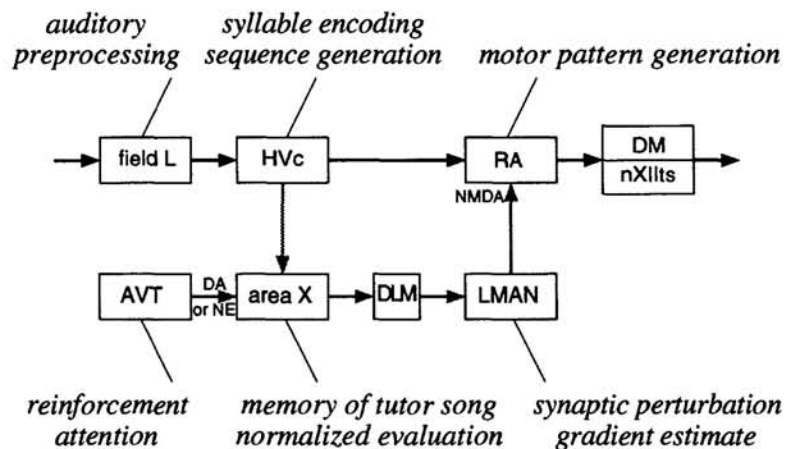

Figure 2: Schematic of primary song control nuclei and their proposed functions in the present model of bird song learning.

in vocalization because lesions in these nuclei in adult birds do not impair their crystallized songs. However, lesions in area X and LMAN during the motor learning phase result in contrasting deficits. The songs of LMAN-lesioned birds crystallize prematurely, whereas the songs of area X-lesioned birds remain variable (Scharff and Nottebohm, 1991). It has been suggested that this pathway is responsible for the storage of song templates (Doupe and Konishi, 1991) or guidance of the synaptic connection from HVc to RA (Mooney, 1992).

## 3   FUNCTIONAL NEUROANATOMY OF BIRDSONG

The song learning process can be decomposed into three stages. In the first stage, suitable internal acoustic representations of syllables and syllable combinations are constructed. This "auditory template" can be assembled by unsupervised learning schemes like clustering and principal components analysis. The second stage involves the encoding of phonetic sequences using the internal representation. If the representation is sparse or nearly orthogonal, sequential transition can be easily encoded by Hebbian learning. The third stage is an inverse mapping from the internal auditory representation into spatio-temporal patterns of motor commands. This can be accomplished by exploration in the space of motor commands using reinforcement learning. The responses of the units that encode the acoustic primitives can be used to the evaluate the resulting auditory signal and direct the exploration.

How are these three computational stages organized within the brain areas and pathways of the songbird? Figure 2 gives an overview of our current working hypothesis. Auditory inputs are pre-processed in field L. Some higher-order representations, such as syllables and syllable combinations, are established in HVc depending on the bird's auditory experience. Moreover, transitions between syllables are encoded in the HVc network. The sequential activation of syllable coding units in HVc are transformed into the time course of motor commands in RA. DM and nXIIts control breathing and the muscles in syrinx, bird's vocal organ.

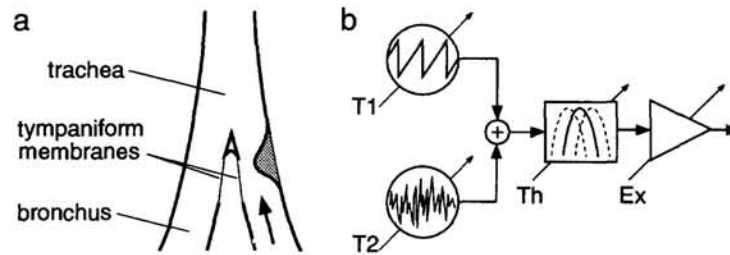

Figure 3: (a) The syrinx of songbirds. (b) The model syrinx.

The consequences of selective lesions of areas in the anterior forebrain pathway (Scharff and Nottebohm, 1991) are consistent with the failures expected for a reinforcement learning system. In particular, we suggest that this pathway serves the function of an adaptive critic with stochastic search elements (Barto et al., 1983). We propose that LMAN perturbs the synaptic connections from HVc to RA and area X regulates LMAN by the song evaluation. Modulation of HVc to RA connection by LMAN is biologically plausible since LMAN input to RA is mediated mainly by NMDA type synapses, which can modulate the amplitude of mainly non-NMDA type synaptic input from HVc (Mooney, 1992).

The assumption that area X provides evaluation is supported by the fact that it receives catecholaminergic projection (dopamine of norepinephrine) from a midbrain nucleus AVT (Lewis et al., 1981). These neurotransmitters are used in many species for reinforcement or attention signals. It is known that auditory learning is enhanced when associated with visual or social interaction with the tutor. Area X is a candidate region where auditory inputs from HVc are associated with reinforcing input from AVT during auditory learning.

## 4   CONSTRUCTION OF SONG LEARNING MODEL

In order to test the above hypothesis, we constructed a computer model of the birdsong learning system. The specific aim was to simulate the process of explorative motor learning, in which the time course of motor command for each syllable is determined by auditory template matching. We assumed that orthogonal representations for syllables and their sequential activation were already established in HVc and that an auditory template matching mechanism exists in area X.

### 4.1   The syrinx

The bird's syrinx is located near the junction of the trachea and the bronchi (Vicario, 1991). Its sound source is the tympaniform membrane which faces to the bronchus on one side and the air sac on the other (Figure 3a). When some of the syringeal muscles contract, the lumen of the bronchus is throttled and produces vibration in the membrane. When stretched along one dimension, the membrane produces harmonic sounds, but when stretched along two dimensions, the sound contains non-harmonic components (Casey and Gaunt, 1985).

Accordingly, we provided two sound sources for the model syrinx (Figure 3b). The fundamental frequency of the harmonic component was controlled by the membrane tension in one direction (T1). The amplitude of the noisy component was proportional to the

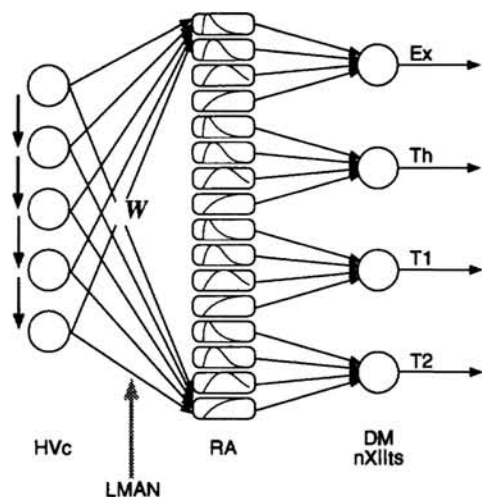

Figure 4: RA units with different spatio-temporal output profiles are driven by locally-coded HVc units. LMAN perturbs the weights $W$ between HVc and RA. The output units in DM and nXIIts drive the model syrinx.

membrane tension in an orthogonal direction (T2). Mixture of these sounds went through a bandpass filter whose resonance frequency was controlled by the throttling of the bronchus (Th). The overall sound amplitude was determined by the strength of expiration (Ex). By controlling the time course of these four variables (Ex,Th,T1,T2), the model could produce wide variety of "bird-like" chirps and warbles.

## 4.2   Motor pattern generation in RA

RA is topographically organized, each part projecting to different motoneuron pools in nXIIts (Vicario, 1991). Also, RA neurons have complex temporal responses to the inputs from HVc (Mooney, 1992). Therefore, we assumed that RA consists of groups of neurons with specific spatial and temporal output properties, as shown in Figure 4. For each of the four motor command variables, we provided several units with different temporal response kernels. The sequential activation of syllable coding units in HVc drove the RA units through the weights $W$. Their responses were linearly combined and squashed between 0 and 1 to make the final motor commands.

## 4.3   Weight space search by LMAN and area X

With the above model of the motor output, the task is to find a connection matrix $W$ that maximizes a template matching measure. One way for doing this is to perturb the output of the units and correlate it with the input and the evaluation (Barto et al., 1983). An alternative way, adopted here, is to perturb the weights and correlate them with the evaluation.

We used the following stochastic gradient ascent algorithm. The weight matrix $W$ is modulated by $\Delta W$, given by the sum of the evaluation gradient estimate $G$ and a random component. The modulated weight persists if the resulting vocalization is better than the recent average evaluation $E[r]$. The evaluation gradient estimate $G$ is updated by the sum

of the perturbations $\Delta W$ weighted with the normalized evaluation.

$$\Delta W := G + \text{random perturbation}$$

$$r := \text{evaluation of the song generated with } W + \Delta W$$

$$W := W + \Delta W \qquad \text{if } r > E[r]$$

$$G := \alpha \frac{r - E[r]}{\sqrt{V[r]}} \Delta W + (1 - \alpha)G,$$

where $0 < \alpha < 1$ provides a form of "momentum" in weight space similar to that used in supervised learning. The average and the variance of evaluation are also estimated on-line as follows.

$$E[r] := \alpha r + (1 - \alpha)E[r]$$

$$V[r] := \alpha(r - E[r])^2 + (1 - \alpha)V[r].$$

### 4.4   Spectrographic template matching

We assumed that evaluation for vocalization is given separately for each of the syllables in a song. The sound signal was analyzed by an 80 channel spectrogram. Each output channel was sent to an analog delay line similar to the gamma filter (de Vries and Principe, 1992). The snapshot image of this (80 channels) × (12 steps) delay line at the end of each syllable was stored as the template. The same delay line image of the syllable generated by the model was compared with the template. This allowed some compensation for variable syllable duration. The direction cosine between the two delay line images was used for the reinforcement signal, $r$.

## 5   SIMULATION RESULTS

One phrase of a recorded zebra finch song (Figure 5a) was the target. Templates were stored for the five syllables in the phrase. Five HVc units coded the five different syllables and 16 RA units represented the four output variables and four different temporal kernels. Learning was started with small random weights. After 200 to 300 trials, the syllable evaluation by direction cosine reached 0.9 (Figure 5d, solid line). The syllables of the learned song resembled the overall frequency profiles of the original syllables. The complex spectrographic structure of the original syllables were, however, not accurate (Figure 5b).

One reason for this imperfect replication could be the difference between the vocal organs of the real zebra finch syrinx and our model syrinx. In order to check the significance of this difference, we took syllable templates from the model song (Figure 5b) and trained another model with random initial weights. In this case, the direction cosine went up to 0.96 (Figure 5d, dotted line) and they sounded quite similar to human ears (Figure 5c).

We also checked the importance of the gradient estimate $G$ in our algorithm. The dashed line in Figure 5d shows the performance of the model with $G \equiv 0$: a simple random walk learning. The learning was hopelessly slow and resembles the deficit seen after lesion of area X (Figure 2).

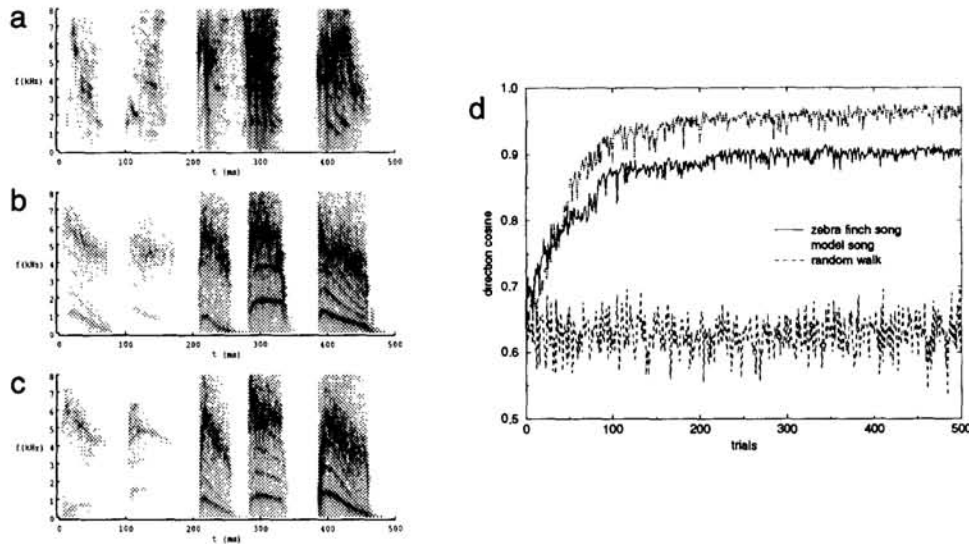

Figure 5: Spectrograms of (a) the original zebra finch song, (b) the learned song based on the tutor in (a), and (c) the second generation learned song based on the tutor in (b). (d) Learning curves for two tutors: a zebra finch song (solid line) and a model song (dotted line) compared with an undirected search in weight space (dashed line). Weight perturbation was given by a Gaussian distribution with $\sigma = 0.1$. The averaging parameter was $\alpha = 0.2$. Simulating 500 trials took 30 minutes on Sparc Station 10.

## 6 Discussion

We have assumed that each vocalized syllable was separately evaluated. If the evaluation is given only at the end of one song or a phrase, learning can be much more difficult because of the temporal credit assignment problem. If we assume that birds take the easiest strategy available, there should be syllable specific evaluation and separate perturbation mechanisms. In some songbirds, individual syllables are practiced out of order at an early stage, and only later is the sequence matched to the auditory template.

Selectivity of auditory responses in both HVc and area X develop during motor learning (Volman, 1993; Doupe, 1993). We can expect such change in response tuning in area X if the evaluations of syllables or syllable sequences are normalized with respect to recent average performance, as we assumed in our model.

Many simplifying assumptions were made in the present model: syllables were unary coded in HVc; simple spectrographic template matching was used; the number of motor output variables and temporal kernels were fairly small; and the sound synthesizer was much simpler than a real syrinx. However, it is not difficult to replace these idealizations with more biologically accurate models. Since the number of learning trials needed in the present model was much less than in the real birdsong learning (tens of thousands of trials), there is margin for further elaboration.

## Acknowledgments

We thank M. Lewicki for the zebra finch song data and M. Konishi, A. Doupe, M. Lewicki, E. Vu, D. Perkel and G. Striedter for their helpful discussions.

## References

Barto, A. G., Sutton, R. S., and Anderson, C. W. (1983). Neuronlike adaptive elements that can solve difficult learning control problems. *IEEE Transactions on System, Man, and Cybernetics*, SMC-13:834–846.

Casey, R. M. and Gaunt, A. S. (1985). Theoretical models of the avian syrinx. *Journal of Theoretical Biology*, 116:45–64.

de Vries, B. and Principe, J. C. (1992). The gamma model—A new neural model for temporal processing. *Neural Networks*, 5:565–576.

Doupe, A. J. (1993). A neural circuit specialized for vocal learning. *Current Opinion in Neurobiology*, 3:104–111.

Doupe, A. J. and Konishi, M. (1991). Song-selective auditory circuits in the vocal control system of the zebra finch. *Proceedings of the National Academy of Sciences, USA*, 88:11339–11343.

Harris-Warrick, R. M., Marder, E., Selverston, A. I., and Moulins, M. (1992). *Dynamic Biological Networks—The Stomatogastric Nervous System*. MIT Press, Cambridge, MA.

Konishi, M. (1965). The role of auditory feedback in the control of vocalization in the white-crowned sparrow. *Zeitschrift fur Tierpsychologie*, 22:770–783.

Lewis, J. W., Ryan, S. M., Arnold, A. P., and Butcher, L. L. (1981). Evidence for a catecholaminergic projection to area x in the zebra finch. *Journal of Comparative Neurology*, 196:347–354.

Mooney, R. (1992). Synaptic basis of developmental plasticity in a birdsong nucleus. *Journal of Neuroscience*, 12:2464–2477.

Nottebohm, F. (1991). Reassessing the mechanisms and origins of vocal learning in birds. *Trends in Neurosciences*, 14:206–211.

Scharff, C. and Nottebohm, F. (1991). A comparative study of the behavioral deficits following lesions of various parts of the zebra finch song systems: Implications for vocal learning. *Journal of Neuroscience*, 11:2896–2913.

Vicario, D. S. (1991). Neural mechanisms of vocal production in songbirds. *Current Opinion in Neurobiology*, 1:595–600.

Volman, S. F. (1993). Development of neural selectivity for birdsong during vocal learning. *Journal of Neuroscience*, 13:4737–4747.

Vu, E. T., Mazurek, M. E., and Kuo, Y.-C. (1994). Identification of a forebrain motor programming network for the learned song of zebra finches. *Journal of Neuroscience*, 14:6924–6934.